# On input selection with reversible jump Markov chain Monte Carlo sampling

**Peter Sykacek**
Austrian Research Institute for Artificial Intelligence (ÖFAI)
Schottengasse 3, A-1010 Vienna, Austria
*peter@ai.univie.ac.at*

## Abstract

In this paper we will treat input selection for a radial basis function (RBF) like classifier within a Bayesian framework. We approximate the a-posteriori distribution over both model coefficients and input subsets by samples drawn with Gibbs updates and reversible jump moves. Using some public datasets, we compare the classification accuracy of the method with a conventional ARD scheme. These datasets are also used to infer the a-posteriori probabilities of different input subsets.

## 1  Introduction

Methods that aim to determine relevance of inputs have always interested researchers in various communities. Classical feature subset selection techniques, as reviewed in [1], use search algorithms and evaluation criteria to determine *one* optimal subset. Although these approaches can improve classification accuracy, they do not explore different equally probable subsets. Automatic relevance determination (ARD) is another approach which determines relevance of inputs. ARD is due to [6] who uses Bayesian techniques, where hierarchical priors penalize irrelevant inputs.

Our approach is also "Bayesian": Relevance of inputs is measured by a probability distribution over all possible feature subsets. This probability measure is determined by the Bayesian evidence of the corresponding models. The general idea was already used in [7] for variable selection in linear regression models. Though our interest is different as we select inputs for a nonlinear classification model. We want an approximation of the true distribution over *all* different subsets. As the number of subsets grows exponentially with the total number of inputs, we can not calculate Bayesian model evidence directly. We need a method that samples efficiently across different dimensional parameter spaces. The most general method that can do this is the reversible jump Markov chain Monte Carlo sampler (reversible jump MC) recently proposed in [4]. The approach was successfully applied by [8] to determine a probability distribution in a mixture density model with variable number of kernels and in [5] to sample from the posterior of RBF regression networks with variable number of kernels. A Markov chain that switches between different input subsets is useful for two tasks: Counting how often a particular subset was visited gives us a relevance measure of the corresponding inputs; For classification, we approximate

the integral over input sets and coefficients by summation over samples from the Markov chain.

The next sections will show how to implement such a reversible jump MC and apply the proposed algorithm to classification and input evaluation using some public datasets. Though the approach could not improve the MLP-ARD scheme from [6] in terms of classification accuracy, we still think that it is interesting: We can assess the importance of different *feature subsets* which is different than importance of *single features* as estimated by ARD.

## 2 Methods

The classifier used in this paper is a RBF like model. Inference is performed within a Bayesian framework. When conditioning on *one* set of inputs, the posterior over model parameters is already multimodal. Therefore we resort to Markov chain Monte Carlo (MCMC) sampling techniques to approximate the desired posterior over both model coefficients and feature subsets. In the next subsections we will propose an appropriate architecture for the classifier and a *hybrid sampler* for model inference. This hybrid sampler consists of two parts: We use Gibbs updates ([2]) to sample when conditioning on a particular set of inputs and reversible jump moves that carry out dimension switching updates.

### 2.1 The classifier

In order to allow input relevance determination by Bayesian model selection, the classifier needs at least one coefficient that is associated with each input: Roughly speaking, the probability of each model is proportional to the likelihood of the most probable coefficients, weighted by their posterior width divided by their prior width. The first factor always increases when using more coefficients (or input features). The second will decrease the more inputs we use and together this gives a peak for the most probable model. A classifier that satisfies these constraints is the so called *classification in the sampling paradigm*. We model class conditional densities and together with class priors express posterior probabilities for classes. In neural network literature this approach was first proposed in [10]. We use a model that allows for overlapping class conditional densities:

$$p(\underline{x}|k) = \sum_{d=1}^{D} w_{kd} p(\underline{x}|\underline{\Phi}_d) \ , \ p(\underline{x}) = \sum_{k=1}^{K} P_k p(\underline{x}|k) \tag{1}$$

Using $P_k$ for the $K$ class priors and $p(\underline{x}|k)$ for the class conditional densities, (1) expresses posterior probabilities for classes as $P(k|\underline{x}) = P_k p(\underline{x}|k)/p(\underline{x})$. We choose the component densities, $p(\underline{x}|\underline{\Phi}_d)$, to be Gaussian with restricted parametrisation: Each kernel is a multivariate normal distribution with a mean and a diagonal covariance matrix. For all Gaussian kernels together, we get $2 * D * I$ parameters, with $I$ denoting the current input dimension and $D$ denoting the number of kernels. Apart from kernel coefficients, $\Phi_d$, (1) has $D$ coefficients per class, $w_{kd}$, indicating the prior kernel allocation probabilities and $K$ class priors. Model (1) allows to treat labels of patterns as *missing data* and use labeled as well as unlabeled data for model inference. In this case training is carried out using the likelihood of observing inputs *and* targets:

$$p(\mathcal{T}, \mathcal{X}|\underline{\Theta}) = \Pi_{k=1}^{K} \Pi_{n_k=1}^{N_k} P_k p_k(\underline{x}_{n_k}|\underline{\Theta}_k) \Pi_{m=1}^{M} p(\underline{x}_m|\underline{\Theta}), \tag{2}$$

where $\mathcal{T}$ denotes labeled and $\mathcal{X}$ unlabeled training data. In (2) $\underline{\Theta}_k$ are all coefficients the $k$-th class conditional density depends on. We further use $\underline{\Theta}$ for all model

coefficients together, $n_k$ as number of samples belonging to class $k$ and $m$ as index for unlabeled samples. To make Gibbs updates possible, we further introduce two latent allocation variables. The first one, $d$, indicates the kernel number each sample was generated from, the second one is the unobserved class label $c$, introduced for unlabeled data. Typical approaches for training models like (1), e.g. [3] and [9], use the EM algorithm, which is closely related to the Gibbs sampler introduce in the next subsection.

## 2.2  Fixed dimension sampling

In this subsection we will formulate Gibbs updates for sampling from the posterior when conditioning on a fixed set of inputs. In order to allow sampling from the full conditional, we have to choose priors over coefficients from their *conjugate family*:

- Each component mean, $\underline{m}_d$, is given a Gaussian prior: $\underline{m}_d \sim \mathcal{N}_d(\underline{\xi}, \underline{\kappa})$.

- The inverse variance of input $i$ and kernel $d$ gets a Gamma prior: $\sigma_{id}^{-2} \sim \Gamma(\alpha, \beta_i)$.

- All $d$ variances of input $i$ have a common hyperparameter, $\beta_i$, that has itself a Gamma hyperprior: $\beta_i \sim \Gamma(g, h_i)$.

- The mixing coefficients, $\underline{w}_k$, get a Dirichlet prior: $\underline{w}_k \sim \mathcal{D}(\delta_w, ..., \delta_w)$.

- Class priors, $\underline{P}$, also get a Dirichlet prior: $\underline{P} \sim \mathcal{D}(\delta_P, ..., \delta_P)$.

The quantitative settings are similar to those used in [8]: Values for $\alpha$ are between 1 and 2, $g$ is usually between 0.2 and 1 and $h_i$ is typically between $1/R_i^2$ and $10/R_i^2$, with $R_i$ denoting the $i$'th input range. The mean gets a Gaussian prior centered at the midpoint, $\underline{\xi}$, with diagonal inverse covariance matrix $\underline{\kappa}$, with $\kappa_{ii} = 1/R_i^2$. The prior counts $\delta_w$ and $\delta_P$ are set to 1 to give the corresponding probabilities non-informative proper Dirichlet priors.

The Gibbs sampler uses updates from the full conditional distributions in (3). For notational convenience we use $\Theta_k$ for the parameters that determine class conditional densities. We use $m$ as index over unlabeled data and $c_m$ as latent class label. The index for all data is $n$, $d_n$ are the latent kernel allocations and $n_d$ the number of samples allocated by the $d$-th component. One distribution does not occur in the prior specification. That is $\mathcal{M}n(1, ...)$ which is a multinomial-one distribution. Finally we need some counters: $m_1 ... m_K$ are the counts per class and $m_{1k} .. m_{Dk}$ count kernel allocations of class-$k$-patterns. The full conditional of the $d$-th kernel variances and the hyper parameter $\beta_i$ contain $i$ as index of the input dimension. There we express each $\sigma_{i,d}^{-2}$ separately. In the expression of the $d$-th kernel mean,

$\underline{m}_d$, we use $\underline{V}_d$ to denote the entire covariance matrix.

$$
\begin{aligned}
p(c_m|...) &= \mathcal{M}n\left(1,\left\{\frac{P_k p(\underline{x}_m|\underline{\Theta}_k)}{\sum_k P_k p(\underline{x}_m|\underline{\Theta}_k)}, k=1..K\right\}\right) \quad (3)\\[2mm]
p(d_n|...) &= \mathcal{M}n\left(1,\left\{\frac{w_{t_n d}p(\underline{x}_n|\underline{\Phi}_d)}{\sum_l w_{t_n d}p(\underline{x}_n|\underline{\Phi}_d)}, d=1..D\right\}\right)\\[2mm]
p(\underline{\beta}_i|...) &= \Gamma\left(g+D\alpha, h_i+\sum_d \sigma_{d,i}^{-2}\right)\\[2mm]
p(\underline{w}_k|...) &= \mathcal{D}\left(\delta_w+m_{1k},...,\delta_w+m_{Dk}\right)\\
p(\underline{P}|...) &= \mathcal{D}\left(\delta_P+m_1,...,\delta_P+m_K\right)\\
p(\underline{m}_d|...) &= \mathcal{N}\left((n_d\underline{V}_d^{-1}+\underline{\kappa})^{-1}(n_d\underline{V}_d^{-1}\bar{\underline{x}}_d+\underline{\kappa}\underline{\xi}),(n_d\underline{V}_d^{-1}+\underline{\kappa})^{-1}\right)\\[2mm]
p(\sigma_{i,d}^{-2}|...) &= \Gamma\left(\alpha+\frac{n_d}{2},\beta_i+\frac{1}{2}\sum_{\underline{x}_n \forall n|d_n=d}(\underline{x}_{n,i}-\underline{m}_{d,i})^2\right)
\end{aligned}
$$

## 2.3 Moving between different input subsets

The core part of this sampler are reversible jump updates, where we move between different feature subsets. The probability of a feature subset will be determined by the corresponding Bayesian model evidence and by an additional prior over number of inputs. In accordance with [7], we use the truncated Poisson prior:

$$p(I)=1/\binom{I_{max}}{I}c\frac{\lambda^I}{I!}, \text{ where } c \text{ is a constant and } I_{max} \text{ the total nr. of inputs.}$$

Reversible jump updates are generalizations of conventional Metropolis-Hastings updates, where moves are bijections $(x,u) \leftrightarrow (x',u')$. For a thorough treatment we refer to [4]. In order to switch subsets efficiently, we will use two different types of moves. The first consist of a step where we add one input chosen at random and a matching step that removes one randomly chosen input. A second move exchanges two inputs which allows "tunneling" through low likelihood areas.

Adding an input, we have to increase the dimension of all kernel means and diagonal covariances. These coefficients are drawn from their priors. In addition the move proposes new allocation probabilities in a semi deterministic way. Assuming the ordering, $w_{k,d} \leq w_{k,d+1}$:

$$
\begin{aligned}
\delta_p &= \text{Beta}(b_a,b_b+I)\\
\forall d \leq D/2 \quad &\begin{cases} w'_{k,D+1-d}=w_{k,D+1-d}+w_{k,d}\delta_p \\ w'_{k,d}=w_{k,d}(1-\delta_p) \end{cases} \quad (4)
\end{aligned}
$$

The matching step proposes removing a randomly chosen input. Removing corresponding kernel coefficients is again combined with a semi deterministic proposal of new allocation probabilities, which is exactly symmetric to the proposal in (4).

Table 1: Summary of experiments

| Data | avg(#) | max(#) | RBF (%,$n_a$) | MLP (%,$n_b$) |
|------|--------|--------|---------------|---------------|
| Ionosphere | 4.3 | 9 | (91.5,11) | (95.5,4) |
| Pima | 4 | 7 | (78.9,11) | (79.8,8) |
| Wine | 4.4 | 8 | (100, 0) | (96.8,2) |

We accept births with probability:

$$
\begin{aligned}
\alpha_b \;=\; & \min(1, \text{lh. rt.} \times \frac{p(I+1)}{p(I)} \left(\frac{1}{R'}\sqrt{2\pi}\right)^D \prod_D \exp\left(-0.5\frac{1}{R'^2}(\mu'_d - \xi'_d)^2\right) \\
& \times \left(\frac{\beta'^\alpha}{\Gamma(\alpha)}\right)^D \prod_D (\sigma_d'^{-2})^{\alpha-1} \exp(-\beta'\sigma_d'^{-2}) \\
& \times \frac{d_m/(I+1)}{b_m/(I_{max}-I)} \times \frac{1}{\left(\frac{1}{R'}\sqrt{2\pi}\right)^D \prod_D \exp\left(-0.5\frac{1}{R'^2}(\mu'_d - \xi'_d)^2\right)} \\
& \times \frac{1}{\left(\frac{\beta'^\alpha}{\Gamma(\alpha)}\right)^D \prod_D (\sigma_d'^{-2})^{\alpha-1} \exp(-\beta'\sigma_d'^{-2})}).
\end{aligned}
\tag{5}
$$

The first line in (5) are the likelihood and prior ratio. The prior ratio results from the difference in input dimension, which affects the kernel means and the prior over number of inputs. The first term of the proposal ratio is from proposing to add or remove one input. The second term is the proposal density of the additional kernel components which cancels with the corresponding term in the prior ratio. Due to symmetry of the proposal (4) and its reverse in a death move, there is no contribution from changing allocation probabilities. Death moves are accepted with probability $\alpha_d = 1/\alpha_b$.

The second type of move is an exchange move. We select a new input and one from the model inputs and propose new mean coefficients. This gives the following acceptance probability:

$$
\begin{aligned}
\alpha_c \;=\; & min(1, \text{lh. ratio} \times \frac{\left(\frac{1}{R'}\sqrt{2\pi}\right)^D \prod_D \exp\left(-0.5\frac{1}{R'^2}(\mu'_d - \xi'_d)^2\right)}{\left(\frac{1}{R'}\sqrt{2\pi}\right)^D \prod_D \exp\left(-0.5\frac{1}{R'^2}(\mu_d - \xi_d)^2\right)} \\
& \times \frac{c_m/I}{c_m/(I_{max}-I)} \times \frac{\prod_D \mathcal{N}(\mu_d|...)}{\prod_D \mathcal{N}(\mu'_d|...)}).
\end{aligned}
\tag{6}
$$

The first line of (6) are again likelihood and prior ratio. For exchange moves, the prior ratio is just the ratio from different values in the kernel means. The first term in the proposal ratio is from proposing to exchange an input. The second term is the proposal density of new kernel mean components. The last part is from proposing new allocation probabilities.

## 3    Experiments

Although the method can be used with labeled and unlabeled data, the following experiments were performed using only labeled data. For all experiments we set $\alpha = 2$ and $g = 0.2$. The first two data sets are from the UCI repository[1]. We use

the Ionosphere data which has 33 inputs, 175 training and 176 test samples. For this experiment we use 6 kernels and set $h = 0.5$. The second data is the wine recognition data which provides 13 inputs, 62 training and 63 test samples. For this data, we use 3 kernels and set $h = 0.28$. The third experiment is performed with the Pima data provided by B. D. Ripley[2]. For this one we use 3 kernels and set $h = 0.16$.

For all experiments we draw 15000 samples from the posterior over coefficients and input subsets. We discard the first 5000 samples as burn in and use the rest for predictions. Classification accuracy, is compared with an MLP classifier using R. Neals hybrid Monte Carlo sampling with ARD priors on inputs. These experiments use 25 hidden units. Table 1 contains further details: avg(#) is the average and max(#) the maximal number of inputs used by the hybrid sampler; RBF (%, $n_a$) is the classification accuracy of the hybrid sampler and the number of errors it made that were not made by the ARD-MLP; MLP(%, $n_b$) is the same for the ARD-MLP. We compare classifiers by testing $(n_a, n_b)$ against the null hypothesis that this is an observation from a Binomial $\mathcal{B}n(n_a + n_b, 0.5)$ distribution. This reveals that neither difference is significant. Although we could not improve classification accuracy on these data, this does not really matter because ARD methods usually lead to high generalization accuracy and we can compete.

The real benefit from using the hybrid sampler is that we can infer probabilities telling us how much different subsets contribute to an explanation of the target variables. Figure 3 shows the occurrence probabilities of feature subsets and features. Note that table 1 has also details about how many features were used in these problems. Especially the results from Ionosphere data are interesting as on average we use only 4.3 out of 33 input features. For ionosphere and wine data the Markov chain visits about 500 different input subsets within 10000 samples. For the Pima data the number is about 60 and an order of magnitude smaller.

# 4 Discussion

In this paper we have discussed a hybrid sampler that uses Gibbs updates and reversible jump moves to approximate the a-posteriori distribution over parameters and input subsets in nonlinear classification problems. The classification accuracy of the method could compete with R. Neals MLP-ARD implementation. However the real advantage of the method is that it provides us with a relevance measure of feature subsets. This allows to infer the optimal number of inputs and how many different explanations the data provides.

### Acknowledgements

I want to thank several people for having used resources they provide: I have used R.Neals hybrid Markov chain sampler for the MLP experiments; The data used for the experiments were obtained form the University at Irvine repository and from B. D. Ripley. Furthermore I want to express gratitude to the anonymous reviewers for their comments and to J.F.G. de Freitas for useful discussions during the conference. This work was done in the framework of the research project GZ 607.519/2-V/B/9/98 "Verbesserung der Biosignalverarbeitung durch Beruecksichtigung von Unsicherheit und Konfidenz", funded by the Austrian federal ministry of science and transport (BMWV).

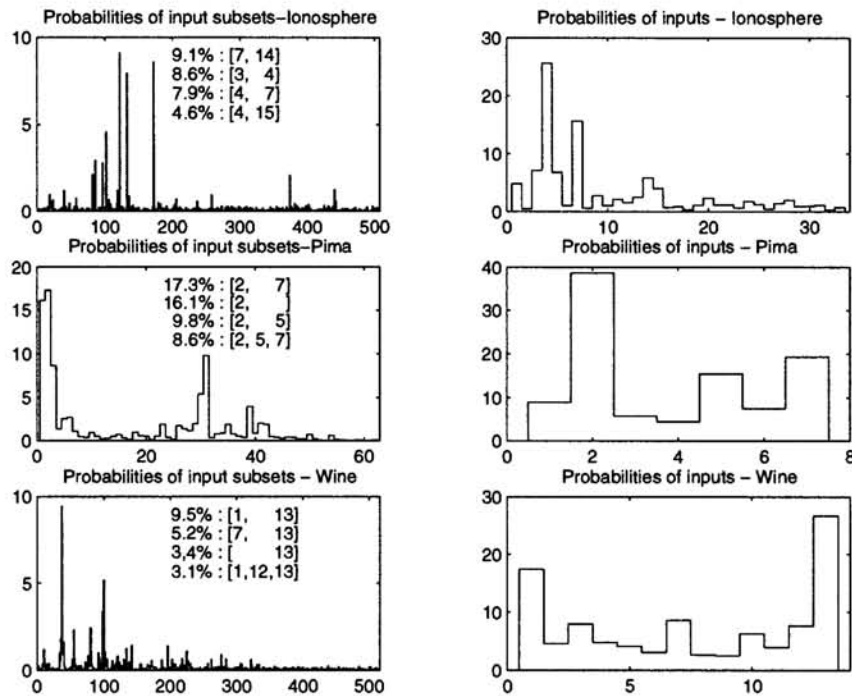

Figure 1: Probabilities of inputs and input subsets measuring their relevance.

## Footnotes

[1] Available at http://www.ics.uci.edu/ mlearn/MLRepository.html.

[2]Available at http://www.stats.ox.ac.uk

# References

[1] P. A. Devijver and J. V. Kittler. *Pattern Recognition. A Statistical Approach.* Prentice-Hall, Englewood Cliffs, NJ, 1982.

[2] S. Geman and D. Geman. Stochastic relaxation, gibbs distributions and the bayesian restoration of images. *IEEE Trans. Pattn. Anal. Mach. Intel.*, 6:721–741, 1984.

[3] Z. Ghahramani, M.I. Jordan Supervised Learning from Incomplete Data via an EM Approach In Cowan J.D., et al.(eds.), Advances in Neural Information Processing Systems 6, Morgan Kaufmann, Los Altos/Palo Alto/San Francisco, pp.120-127, 1994.

[4] P. J. Green. Reversible jump markov chain monte carlo computation and bayesian model determination. *Biometrika*, 82:711–732, 1995.

[5] C. C. Holmes and B. K. Mallick. Bayesian radial basis functions of variable dimension. *Neural Computation*, 10:1217–1234, 1998.

[6] R. M. Neal. Bayesian Learning for Neural Networks. Springer, New York, 1986.

[7] D. B. Phillips and A. F. M. Smith. Bayesian model comparison via jump diffusioons. In W.R. Gilks, S. Richardson, and D.J. Spiegelhalter, editors, *Markov Chain Monte Carlo in Practice*, pages 215–239, London, 1996. Chapman & Hall.

[8] S. Richardson and P.J. Green On Bayesian Analysis of Mixtures with an unknown number of components *Journal Royal Stat. Soc. B*, 59:731–792, 1997.

[9] M. Stensmo, T.J. Sejnowski A Mixture Model System for Medical and Machine Diagnosis In Tesauro G., et al.(eds.), Advances in Neural Information Processing System 7, MIT Press, Cambridge/Boston/London, pp.1077-1084, 1995.

[10] H. G. C. Tråvén A neural network approach to statistical pattern classification by "semiparametric" estimation of probability density functions IEEE Trans. Neur. Net., 2:366–377, 1991.